# Translated Learning: Transfer Learning across Different Feature Spaces

[†]**Wenyuan Dai,** [†]**Yuqiang Chen,** [†]**Gui-Rong Xue,** [‡]**Qiang Yang** and [†]**Yong Yu**

[†]Shanghai Jiao Tong University
Shanghai 200240, China
{dwyak,yuqiangchen,grxue,yyu}@apex.sjtu.edu.cn

[‡]Hong Kong University of Science and Technology
Kowloon, Hong Kong
qyang@cse.ust.hk

## Abstract

This paper investigates a new machine learning strategy called *translated learning*. Unlike many previous learning tasks, we focus on how to use labeled data from one feature space to enhance the classification of other entirely different learning spaces. For example, we might wish to use labeled text data to help learn a model for classifying image data, when the labeled images are difficult to obtain. An important aspect of translated learning is to build a "bridge" to link one feature space (known as the "source space") to another space (known as the "target space") through a translator in order to migrate the knowledge from source to target. The translated learning solution uses a language model to link the class labels to the features in the source spaces, which in turn is *translated* to the features in the target spaces. Finally, this chain of linkages is completed by tracing back to the instances in the target spaces. We show that this path of linkage can be modeled using a Markov chain and risk minimization. Through experiments on the text-aided image classification and cross-language classification tasks, we demonstrate that our translated learning framework can greatly outperform many state-of-the-art baseline methods.

## 1   Introduction

Traditional machine learning relies on the availability of a large amount of labeled data to train a model in the same feature space. However, labeled data are often scarce and expensive to obtain. In order to save much labeling work, various machine learning strategies have been proposed, including semi-supervised learning [13], transfer learning [3, 11, 10], self-taught learning [9], etc. One commonality among these strategies is they all require the training data and test data to be in the same feature space. For example, if the training data are documents, then the classifiers cannot accept test data from a video space. However, in practice, we often face the problem where the labeled data are scarce in its own feature space, whereas there are sufficient labeled data in other feature spaces. For example, there may be few labeled images available, but there are often plenty of labeled text documents on the Web (e.g., through the Open Directory Project, or ODP, http://www.dmoz.org/). Another example is cross-language classification where labeled documents in English are much more than ones in some other languages such as Bangla, which has only 21 Web pages in the ODP. Therefore, it would be great if we could learn the knowledge across different feature spaces and to save a substantial labeling effort.

To address the transferring of knowledge across different feature spaces, researchers have proposed multi-view learning [2, 8, 7] in which each instance has multiple views in different feature spaces. Different from multi-view learning, in this paper, we focus on the situation where the training data are in a *source* feature space, and the test data are in a different *target* feature space, and that there is no correspondence between instances in these spaces. The source and target feature spaces can be

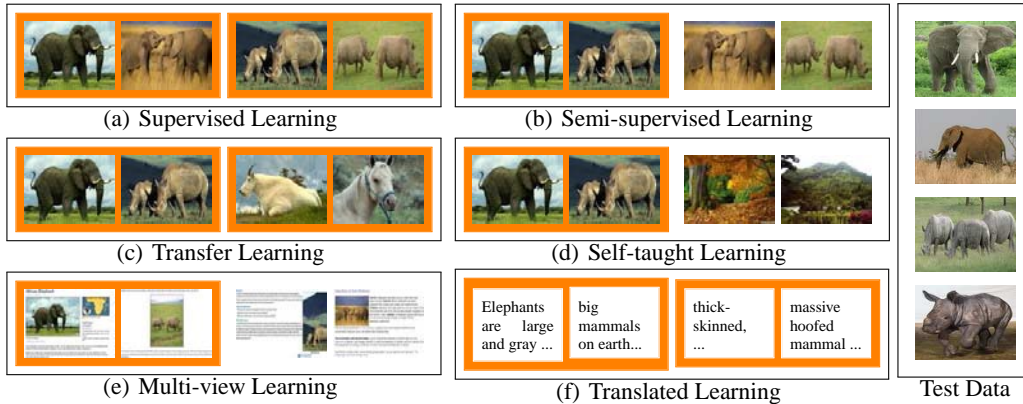

Figure 1: An intuitive illustration to different kinds of learning strategies using classification of image elephants and rhinos as the example. The images in orange frames are labeled data, while the ones without frames are unlabeled data.

very different, as in the case of text and images. To solve this novel learning problem, we develop a novel framework named as *translated learning*, where training data and test data can be in totally different feature spaces. A translator is needed to be exploited to link the different feature spaces. Clearly, the translated learning framework is more general and difficult than traditional learning problems. Figure 1 presents an intuitive illustration of six different learning strategies, including supervised learning, semi-supervised learning [13], transfer learning [10], self-taught learning [9], multi-view learning [2], and finally, translated learning.

An intuitive idea for translated learning is to somehow *translate* all the training data into a target feature space, where learning can be done within a single feature space. This idea has already been demonstrated successful in several applications in cross-lingual text classification [1]. However, for the more general translated learning problem, this idea is hard to be realized, since machine translation between different feature spaces is very difficult to accomplish in many non-natural language cases, such as translating documents to images. Furthermore, while a text corpus can be exploited for cross-langauge translation, for translated learning, the learning of the "feature-space translator" from available resources is a key issue.

Our solution is to make the best use of available data that have both features of the source and target domains in order to construct a translator. While these data may not be sufficient in building a good classifier for the target domain, as we will demonstrate in our experimental study in the paper, by leveraging the available labeled data in the source domain, we can indeed build effective translators. An example is to translate between the text and image feature spaces using the social tagging data from Web sites such as Flickr (http://www.flickr.com/).

The main contribution of our work is to combine the feature translation and the nearest neighbor learning into a unified model by making use of a language model [5]. Intuitively, our model can be represented using a Markov chain $c \rightarrow y \rightarrow x$, where $y$ represents the features of the data instances $x$. In translated learning, the training data $x_s$ are represented by the features $y_s$ in the source feature space, while the test data $x_t$ are represented by the features $y_t$ in the target feature space. We model the learning in the source space through a Markov chain $c \rightarrow y_s \rightarrow x_s$, which can be connected to another Markov chain $c \rightarrow y_t \rightarrow x_t$ in the target space. An important contribution of our work then is to show how to connect these two paths, so that the new chain $c \rightarrow y_s \rightarrow y_t \rightarrow x_t$, can be used to translate the knowledge from the source space to the target one, where the mapping $y_s \rightarrow y_t$ is acting as a feature-level translator. In our final solution, which we call TLRisk, we exploit the risk minimization framework in [5] to model translated learning. Our framework can accept different distance functions to measure the relevance between two models.

## 2 Translated Learning Framework

### 2.1 Problem Formulation

We first define the translated learning problem formally. Let $\mathcal{X}_s$ be the *source* instance space. In this space, each instance $x_s \in \mathcal{X}_s$ is represented by a feature vector $(y_s^{(1)}, \ldots, y_s^{(n_s)})$, where $y_s^{(i)} \in \mathcal{Y}_s$

and $\mathcal{Y}_s$ is the source feature space. Let $\mathcal{X}_t$ be the *target* instance space, in which each instance $x_t \in \mathcal{X}_t$ is represented by a feature vector $(y_t^{(1)}, \ldots, y_t^{(n_t)})$, where $y_t^{(i)} \in \mathcal{Y}_t$ and $\mathcal{Y}_t$ is the target feature space. We have a labeled training data set $\mathcal{L}_s = \{(x_s^{(i)}, c_s^{(i)})\}_{i=1}^n$ in the source space, where $x_s^{(i)} \in \mathcal{X}_s$ and $c_s^{(i)} \in \mathcal{C} = \{1, \ldots, |\mathcal{C}|\}$ is the true class-label of $x_s^{(i)}$. We also have another labeled training data set $\mathcal{L}_t = \{(x_t^{(i)}, c_t^{(i)})\}_{i=1}^m$ in the target space, where $x_t^{(i)} \in \mathcal{X}_t$ and $c_t^{(i)} \in \mathcal{C}$. Usually, $m$ is assumed to be small, so that $\mathcal{L}_t$ is not enough to train a reliable prediction model. The unlabeled test data set $\mathcal{U}$ is a set of $k$ examples $\{x_u^{(i)}\}_{i=1}^k$, where $x_u^{(i)} \in \mathcal{X}_t$. Note that $x_s^{(i)}$ is in a different feature space from $x_t^{(i)}$ and $x_u^{(i)}$. For example, $x_s^{(i)}$ may be a text document, while $x_t^{(i)}$ and $x_u^{(i)}$ may be visual images.

To link the two feature spaces, a feature translator $p(y_t|y_s) \propto \phi(y_t, y_s)$ is constructed. However, for ease of explanation, we first assume that the translator $\phi$ is given, and will discuss the derivation of $\phi$ later in this section, based on co-occurrence data. We focus on our main objective in learning, which is to estimate a hypothesis $h_t : \mathcal{X}_t \mapsto \mathcal{C}$ to classify the instances $x_u^{(i)} \in \mathcal{U}$ as accurately as possible, by making use of the labeled training data $\mathcal{L} = \mathcal{L}_s \cup \mathcal{L}_t$ and the translator $\phi$.

## 2.2 Risk Minimization Framework

First, we formulate our objective in terms of how to minimize an expected risk function with respect to the labeled training data $\mathcal{L} = \mathcal{L}_s \cup \mathcal{L}_t$ and the translator $\phi$ by extending the risk minimization framework in [5].

In this work, we use the risk function $R(c, x_t)$ to measure the the risk for classifying $x_t$ to the category $c$. Therefore, to predict the label for an instance $x_t$, we need only to find the class-label $c$ which minimizes the risk function $R(c, x_t)$, so that

$$h_t(x_t) = \arg\min_{c \in \mathcal{C}} R(c, x_t). \tag{1}$$

The risk function $R(c, x_t)$ can be formulate as the *expected loss* when $c$ and $x_t$ are relevant; formally,

$$R(c, x_t) \equiv L(r = 1|c, x_t) = \int_{\Theta_{\mathcal{C}}} \int_{\Theta_{\mathcal{X}_t}} L(\theta_{\mathcal{C}}, \theta_{\mathcal{X}_t}, r = 1) p(\theta_{\mathcal{C}}|c) \, p(\theta_{\mathcal{X}_t}|x_t) \, \mathrm{d}\theta_{\mathcal{X}_t} \, \mathrm{d}\theta_{\mathcal{C}}. \tag{2}$$

Here, $r = 1$ represents the event of "relevant", which means (in Equation (2)) "$c$ and $x_t$ are relevant", or "the label of $x_t$ is $c$". $\theta_{\mathcal{C}}$ and $\theta_{\mathcal{X}_t}$ are the models with respect to classes $\mathcal{C}$ and target space instances $\mathcal{X}_t$ respectively. $\Theta_{\mathcal{C}}$ and $\Theta_{\mathcal{X}_t}$ are two corresponding model spaces involving all the possible models. Note that, in Equation (2), $\theta_{\mathcal{C}}$ only depends on $c$ and $\theta_{\mathcal{X}_t}$ only depends to $x_t$. Thus, we use $p(\theta_{\mathcal{C}}|c)$ to replace $p(\theta_{\mathcal{C}}|c, x_t)$, and use $p(\theta_{\mathcal{X}_t}|x_t)$ to replace $p(\theta_{\mathcal{X}_t}|c, x_t)$. $L(\theta_{\mathcal{C}}, \theta_{\mathcal{X}_t}, r = 1)$ is the loss function with respect to the event of $\theta_{\mathcal{C}}$ and $\theta_{\mathcal{X}_t}$ being relevant. We next address the estimation of the risk function in Equation (2).

## 2.3 Estimation

The risk function in Equation (2) is difficult to estimate, since the sizes of $\Theta_{\mathcal{C}}$ and $\Theta_{\mathcal{X}_t}$ can be exponential in general. Therefore, we have to use approximation for estimating the risk function for efficiency. First of all, the loss function $L(\theta_{\mathcal{C}}, \theta_{\mathcal{X}_t}, r = 1)$ can be formulated using distance functions between the two models $\theta_{\mathcal{C}}$ and $\theta_{\mathcal{X}_t}$, so that $L(\theta_{\mathcal{C}}, \theta_{\mathcal{X}_t}, r = 1) = \alpha\Delta(\theta_{\mathcal{C}}, \theta_{\mathcal{X}_t})$, where $\Delta(\theta_{\mathcal{C}}, \theta_{\mathcal{X}_t})$ is the distance (or dissimilarity) function, e.g. the Kullback-Leibler divergence. Replacing $L(\theta_{\mathcal{C}}, \theta_{\mathcal{X}_t}, r = 1)$ with $\Delta(\theta_{\mathcal{C}}, \theta_{\mathcal{X}_t})$, the risk function is reformulated as

$$R(c, x_t) \propto \int_{\Theta_{\mathcal{C}}} \int_{\Theta_{\mathcal{X}_t}} \Delta(\theta_{\mathcal{C}}, \theta_{\mathcal{X}_t}) p(\theta_{\mathcal{C}}|c) \, p(\theta_{\mathcal{X}_t}|x_t) \, \mathrm{d}\theta_{\mathcal{X}_t} \, \mathrm{d}\theta_{\mathcal{C}}. \tag{3}$$

Since the sizes of $\Theta_{\mathcal{C}}$ and $\Theta_{\mathcal{X}_t}$ are exponential in general, we cannot calculate Equation (3) straightforwardly. In this paper, we approximate the risk function by its value at the posterior mode:

$$R(c, x_t) \approx \Delta(\hat{\theta}_c, \hat{\theta}_{x_t}) p(\hat{\theta}_c|c) p(\hat{\theta}_{x_t}|x_t) \propto \Delta(\hat{\theta}_c, \hat{\theta}_{x_t}) p(\hat{\theta}_c|c), \tag{4}$$

where $\hat{\theta}_c = \arg\max_{\theta_{\mathcal{C}}} p(\theta_{\mathcal{C}}|c)$, and $\hat{\theta}_{x_t} = \arg\max_{\theta_{\mathcal{X}_t}} p(\theta_{\mathcal{X}_t}|x_t)$.

In Equation (4), $p(\hat{\theta}_c|c)$ is the prior probability of $\hat{\theta}_c$ with respect to the target class $c$. This prior can be used to balance the influence of different classes in the class-imbalance case. When we assume there is no prior difference among all the classes, the risk function can be rewritten into

**Algorithm 1** Risk Minimization Algorithm for Translated Learning: (`TLRisk`)

---

**Input:** Labeled training data $\mathcal{L}$ in the source space, unlabeled test data $\mathcal{U}$ in the target space, a translator $\phi$ to link the two feature spaces $\mathcal{Y}_s$ and $\mathcal{Y}_t$ and a dissimilarity function $\Delta(\cdot, \cdot)$.
**Output:** The prediction label $h_t(x_t)$ for each $x_t \in \mathcal{U}$.

**Procedure** `TLRisk_train`
  1: **for** each $c \in \mathcal{C}$ **do**
  2:     Estimate the model $\hat{\theta}_c$ based on Equation (6).
  3: **end for**

**Procedure** `TLRisk_test`
  1: **for** each $x_t \in \mathcal{U}$ **do**
  2:     Estimate the model $\hat{\theta}_{x_t}$ based on Equation (7).
  3:     Predict the label $h_t(x_t)$ for $x_t$ based on Equations (1) and (5).
  4: **end for**

---

$$R(c, x_t) \propto \Delta(\hat{\theta}_c, \hat{\theta}_{x_t}), \tag{5}$$

where $\Delta(\hat{\theta}_c, \hat{\theta}_{x_t})$ denotes the dissimilarity between two models $\hat{\theta}_c$ and $\hat{\theta}_{x_t}$. To achieve this objective, as in [5], we formulate these two models in the target feature space $\mathcal{Y}_t$; specifically, if we use KL divergence as the distance function, $\Delta(\hat{\theta}_c, \hat{\theta}_{x_t})$ can be measured by $\mathrm{KL}(p(\mathcal{Y}_t|\hat{\theta}_c)||p(\mathcal{Y}_t|\hat{\theta}_{x_t}))$.

Our estimation is based on the Markov chain assumption where $\hat{\theta}_c \rightarrow c \rightarrow y_s \rightarrow y_t \rightarrow x_t \rightarrow \hat{\theta}_{x_t}$ and $\hat{\theta}_c \rightarrow c \rightarrow y_t \rightarrow x_t \rightarrow \hat{\theta}_{x_t}$, so that

$$p(y_t|\hat{\theta}_c) = \int_{\mathcal{Y}_s} \sum_{c' \in \mathcal{C}} p(y_t|y_s)p(y_s|c')p(c'|\hat{\theta}_c) \, \mathrm{d}y_s + \lambda \sum_{c' \in \mathcal{C}} p(y_t|c')p(c'|\hat{\theta}_c), \tag{6}$$

where $p(y_t|y_s)$ can be estimated using the translator $\phi$; $p(y_s|c')$ can be estimated based on the statistical observations in the labeled text data set $\mathcal{L}_s$ in the source feature space $\mathcal{Y}_s$; $p(y_t|c')$ can be estimated based on $\mathcal{L}_t$ in the target feature space $\mathcal{Y}_t$; $p(c'|\hat{\theta}_c)$ can be calculated as: $p(c'|\hat{\theta}_c) = 1$ if $c = c'$, and otherwise, $p(c'|\hat{\theta}_c) = 0$; and $\lambda$ is a trade-off parameter which controls the influence of target space labeled data $\mathcal{L}_t$.

For another model $p(\mathcal{Y}_t|\hat{\theta}_{x_t})$, it can be estimated by

$$p(y_t|\hat{\theta}_{x_t}) = \int_{\mathcal{X}_t} p(y_t|x_t')p(x_t'|\hat{\theta}_{x_t}) \, \mathrm{d}x_t', \tag{7}$$

where $p(y_t|x_t')$ can be estimated using the feature extractor in the target feature space $\mathcal{Y}_t$, and $p(x_t'|\hat{\theta}_{x_t})$ can be calculated as $p(x_t'|\hat{\theta}_{x_t}) = 1$ if $x_t' = x_t$; otherwise $p(x_t'|\hat{\theta}_{x_t}) = 0$.

Integrating Equations (1), (5), (6) and (7), our translated learning framework is summarized as algorithm `TLRisk`, an abbreviation for *Translated Learning via Risk Minimization*, which is shown in Algorithm 1.

Considering the computational cost of Algorithm 1, due to the Markov chain assumption, our algorithm `TLRisk` can be implemented using dynamic programming. Therefore, in the worst case, the time complexity of `TLRisk` is $O(|\mathcal{C}||\mathcal{Y}_t| + |\mathcal{Y}_t||\mathcal{Y}_s|)$ in training, and $O(|\mathcal{C}||\mathcal{Y}_t|)$ for predicting an instance. In practice, the data are quite sparse, and good feature mappings (or translator) should also be sparse, otherwise it will consist of many ambiguous cases. Therefore, `TLRisk` can perform much faster than the worst cases generally, and the computational cost of `TLRisk` is linear in the non-zero occurrences in feature mappings.

## 2.4 Translator $\phi$

We now explain in particular how to build the translator $\phi(y_t, y_s) \propto p(y_t|y_s)$ to connect two different feature spaces. As mentioned before, to estimate the translator $p(y_t|y_s)$, we need some co-occurrence data across the two feature spaces: source and target. Formally, we need co-occurrence data in the form of $p(y_t, y_s)$, $p(y_t, x_s)$, $p(x_t, y_s)$, or $p(x_t, x_s)$. In cross-language problems, dictionaries can be considered as data in the form of $p(y_t, y_s)$ (feature-level co-occurrence). On the Web,

| Data Set | Data Size | | | | Data Set | Data Size | | | |
|---|---|---|---|---|---|---|---|---|---|
| | Documents | | Images | | | Documents | | Images | |
| | + | − | + | − | | + | − | + | − |
| horse vs coin | 1610 | 1345 | 270 | 123 | dog vs canoe | 1084 | 1047 | 102 | 103 |
| kayak vs bear | 1045 | 885 | 102 | 101 | greyhound vs cd | 380 | 362 | 94 | 102 |
| electric-guitar vs snake | 335 | 326 | 122 | 112 | stained-glass vs microwave | 331 | 267 | 99 | 107 |
| cake vs binoculars | 265 | 320 | 104 | 216 | rainbow vs sheet-music | 261 | 256 | 102 | 84 |
| laptop vs sword | 210 | 203 | 128 | 102 | tomato vs llama | 175 | 172 | 102 | 119 |
| bonsai vs comet | 166 | 164 | 122 | 120 | frog vs saddle | 150 | 148 | 115 | 110 |

Table 1: The description for each data set. Here, `horse vs coin` indicates all the positive instances are about `horse` while all the negative instances are about `coin`. "+" means positive instance; "−" means negative instances.

social annotations on images (e.g. Flickr, images associated with keywords) and search-engine results in response to queries are examples for correlational data in the forms of $p(y_t, x_s)$ and $p(x_t, y_s)$ (feature-instance co-occurrence). Moreover, multi-view data (e.g. Web pages including both text and pictures) is an example for data in the form of $p(x_t, x_s)$ (instance-level co-occurrence). Where there is a pool of such co-occurrence data available, we can build the translator $\phi$ for estimating the Markov chains in the previous subsections.

In particular, to estimate the translator $\phi$, at first, the feature-instance co-occurrence data ($p(y_t, x_s)$ or $p(x_t, y_s)$) can be used to estimate the probabilities for feature-level co-occurrence $p(y_t, y_s)$; formally, $p(y_t, y_s) = \int_{\mathcal{X}_s} p(y_t, x_s) p(y_s|x_s) \, \mathrm{d}x_s$ and $p(y_t, y_s) = \int_{\mathcal{X}_t} p(x_t, y_s) p(y_t|x_t) \, \mathrm{d}x_t$. The instance-level co-occurrence data can also be converted to feature-level co-occurrence; formally, $p(y_t, y_s) = \int_{\mathcal{X}_t} \int_{\mathcal{X}_s} p(x_t, x_s) p(y_s|x_s) p(y_t|x_t) \, \mathrm{d}x_s \mathrm{d}x_t$. Here, $p(y_s|x_s)$ and $p(y_t|x_t)$ are two feature extractors in $\mathcal{Y}_s$ and $\mathcal{Y}_t$. Using the feature-level co-occurrence probability $p(y_t, y_s)$, we can estimate the translator as $p(y_t|y_s) = p(y_t, y_s) / \int_{\mathcal{Y}_t} p(y'_t, y_s) \mathrm{d}y'_t$.

# 3 Evaluation: Text-aided Image Classification

In this section, we apply our framework `TLRisk` to a text-aided image classification problem, which uses binary labeled text documents as auxiliary data to enhance the image classification. This problem is derived from the application where a user or a group of users may have expressed preferences over some text documents, and we wish to translate these preferences to images for the same group of users. We will show the effectiveness of `TLRisk` on text-aided image classification. Our objective is to demonstrate that even with a small amount of labeled image training data, we can still build a high-quality translated learning solution for image classification by leveraging the text documents, even if the co-occurrence data themselves are not sufficient when directly used for training a classification model in the target space.

## 3.1 Data Sets

The data sets of Caltech-256[1] and Open Directory Project (ODP, `http://www.dmoz.org/`) were used in our evaluation, as the image and text corpora. Our ODP collection was crawled during August 2006, and involves 1,271,106 English Web pages. We generated 12 binary text-to-image classification tasks from the above corpora. The description for each data set is presented in Table 1. The first column presents the name of each data set, e.g. `horse vs coin` indicates all the positive instances are about `horse` while all the negative instances are about `coin`. We collected the corresponding documents from ODP for each category. However, due to space limitation, we omit the detailed ODP directory information with respect to each data set here. In the table, we also listed the data sizes for each task, including documents and images. Generally, the number of documents is much larger than the number of images.

For data preprocessing, the SIFT descriptor [6] was used to find and describe the interesting points in the images, and then clustered the extracted interest points into 800 clusters to obtain the code-book. Based on the code-book, each image can be converted to a corresponding feature vector. For text documents, we first extracted and stemmed all the tokens from the ODP Web pages, and then *information gain* [12] was used to select the most important features for further learning process. We collected the co-occurrence data from a commercial image search engine during April 2008. The collected data are in the form of feature-instance co-occurrence $p(y_s, x_t)$, so that we have to convert them to feature-level co-occurrence $p(y_s, y_t)$ as discussed in Section 2.4.

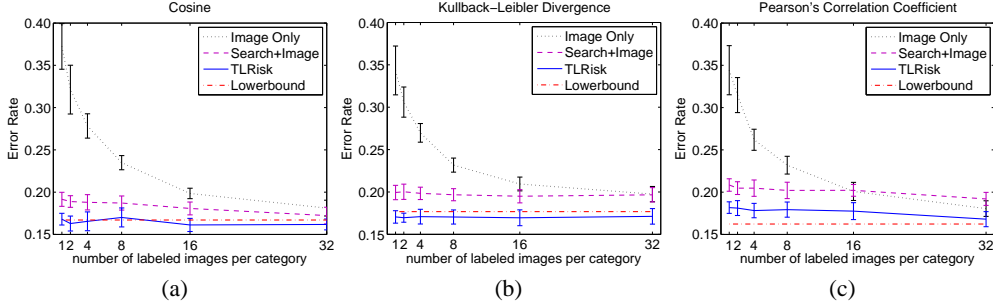

Figure 2: The average error rates over 12 data sets for text-aided image classification with different number of labeled images $\mathcal{L}_t$.

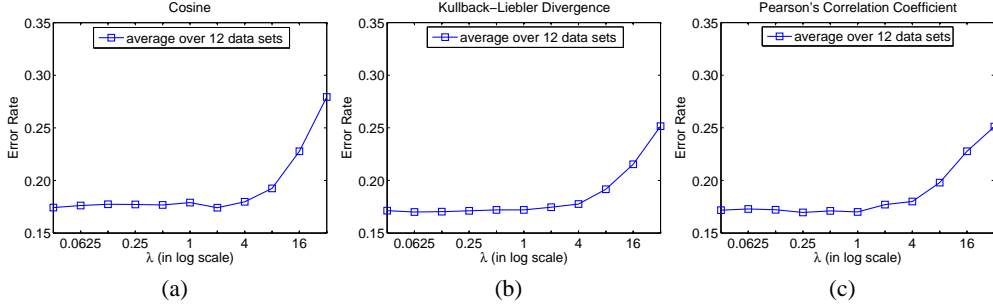

Figure 3: The average error rates over 12 data sets for text-aided image classification with different trade-off $\lambda$.

## 3.2 Evaluation Methods

Few existing research works addressed the text-aided image classification problem, so that for the baseline methods in our experiments, we first simply used the labeled data $\mathcal{L}_t$ as the training data in the target space to train a classification model; we refer to this model as `Image Only`. The second baseline is to use the category name (in this case, there are two names for binary classification problems) to search for training images and then to train classifiers together with labeled images in $\mathcal{L}_t$; we refer to this model as `Search+Image`.

Our framework `TLRisk` was evaluated under three different dissimilarity functions: (1) Kullback-Leibler divergence (named KL): $\int_{\mathcal{Y}_t} p(y_t|\theta_{\mathcal{C}}) \log \frac{p(y_t|\theta_{\mathcal{C}})}{p(y_t|\theta_{\mathcal{X}_t})} \mathrm{d}y_t$; (2) Negative of cosine function (named NCOS): $-\frac{\int_{\mathcal{Y}_t} p(y_t|\theta_{\mathcal{C}}) p(y_t|\theta_{\mathcal{X}_t}) \mathrm{d}y_t}{\sqrt{\int_{\mathcal{Y}_t} p^2(y_t|\theta_{\mathcal{C}}) \mathrm{d}y_t} \sqrt{\int_{\mathcal{Y}_t} p^2(y_t|\theta_{\mathcal{X}_t}) \mathrm{d}y_t}}$; (3) Negative of the Pearson's correlation coefficient (named NPCC): $-\frac{\mathrm{cov}(p(\mathcal{Y}_t|\theta_{\mathcal{C}}), p(\mathcal{Y}_t|\theta_{\mathcal{X}_t}))}{\sqrt{\mathrm{var}(p(\mathcal{Y}_t|\theta_{\mathcal{C}})) \mathrm{var}(p(\mathcal{Y}_t|\theta_{\mathcal{X}_t}))}}$.

We also evaluated the *lower bound* of the error rate with respect to each data set. To estimate the lower bound, we conducted a 5-fold cross-validation on the test data $\mathcal{U}$. Note that this strategy, which is referred to as `Lowerbound`, is unavailable in our problem setting, since it uses a large amount of labeled data in the target space. In our experiments, this lower bound is used just for reference. We also note that on some data sets, the performance of `Lowerbound` may be slightly worse than that of `TLRisk`, because `Lowerbound` was trained based on images in Caltech-256, while `TLRisk` was based on the co-occurrence data. These models used different supervisory knowledge.

## 3.3 Experimental Results

The experimental results were evaluated in terms of error rates, and are shown in Figure 2. On one hand, from the table, we can see that our framework `TLRisk` greatly outperforms the baseline methods `Image Only` and `Search+Image`, no matter which dissimilarity function is applied. On the other hand, compared with `Lowerbound`, `TLRisk` also shows comparable performance. It indicates that our framework `TLRisk` can effectively learn knowledge across different feature spaces in the case of text-to-image classification.

Moreover, when the number of target space labeled images decreases, the performance of `Image Only` declines rapidly, while the performances of `Search+Image` and `TLRisk` stay very sta-

| Data Set | English | | German | |
|---|---|---|---|---|
| | Location | Size | Location | Size |
| 1 | Top: Sport: Ballsport | 2000 | Top: World: Deutsch: Sport: Ballsport | 128 |
| | Top: Computers: Internet | 2000 | Top: World: Deutsch: Computer: Internet | 126 |
| 2 | Top: Arts: Architecture: Building Types | 1259 | Top: World: Deutsch: Kultur: Architektur: Gebäudetypen | 71 |
| | Top: Home: Cooking: Recipe Collections | 475 | Top: World: Deutsch: Zuhause: Kochen: Rezeptesammlungen | 72 |
| 3 | Top: Science: Agriculture | 1886 | Top: World: Deutsch: Wissenschaft: Agrarwissenschaften | 71 |
| | Top: Society: Crime | 1843 | Top: World: Deutsch: Gesellschaft: Kriminalität | 69 |
| 4 | Top: Sports: Skating: Roller Skating | 926 | Top: World: Deutsch: Sport: Rollsport | 70 |
| | Top: Health: Public Health and Safety | 2361 | Top: World: Deutsch: Gesundheit: Public Health | 71 |
| 5 | Top: Recreation: Outdoors: Hunting | 2919 | Top: World: Deutsch: Freizeit: Outdoor: Jagd | 70 |
| | Top: Society: Holidays | 2258 | Top: World: Deutsch: Gesellschaft: Festúnd Feiertage | 72 |

Table 2: The description for each cross-language classification data set.

ble. This indicates that `TLRisk` is not quite sensitive to the size of $\mathcal{L}_t$; in other words, `TLRisk` has good robustness. We also want to note that, sometimes `TLRisk` performs slightly better than `Lowerbound`. This is not a mistake, because these two methods use different supervisory knowledge: `Lowerbound` is based on images in the Caltech-256 corpus; `TLRisk` is based on the co-occurrence data. In these experiments, `Lowerbound` is just for reference.

In `TLRisk`, a parameter to tune is the trade off parameter $\lambda$ (refer to Equation (6)). Figure 3 shows the average error rate curves on all the 12 data sets, when $\lambda$ gradually changes from $2^{-5}$ to $2^5$. In this experiment, we fixed the number of target training images per category to one, and set the threshold $K$ (which is the number of images to collect for each text keyword, when collecting the co-occurrence data) to 40. From the figure, we can see that, on one hand, when $\lambda$ is very large, which means the classification model mainly builds on the target space training images $\mathcal{L}_t$, the performance is rather poor. On the other hand, when $\lambda$ is small such that the classification model relies more on the auxiliary text training data $\mathcal{L}_s$, the classification performance is relatively stable. Therefore, we suggest to set the trade-off parameter $\lambda$ to a small value, and in these experiments, all the $\lambda$s are set to 1, based on Figure 3.

# 4 Evaluation: Cross-language Classification

In this section, we apply our framework `TLRisk` to another scenario, the cross-language classification. We focused on English-to-German classification, where English documents are used as the source data to help classify German documents, which are target data.

In these experiments, we collected the documents from corresponding categories from ODP English pages and ODP German pages, and generated five cross-language classification tasks, as shown in Table 2. For the co-occurrence data, we used the English-German dictionary from the Internet Dictionary Project[2] (IDP). The dictionary data are in the form of feature-level co-occurrence $p(y_t, y_s)$. We note that while most cross-language classification works rely on machine translation [1], our assumption is that the machine translation is unavailable and we rely on dictionary only.

We evaluated `TLRisk` with the negative of cosine (named `NCOS`) as the dissimilarity function. Our framework `TLRisk` was compared to classification using only very few German labeled documents as a baseline, called `German Labels Only`. We also present the lower bound of error rates by performing 5-fold cross-validation on the test data $\mathcal{U}$, which we refer to as `Lowerbound`. The performances of the evaluated methods are presented in Table 3. In this experiment, we have only sixteen German labeled documents in each category. The error rates in Table 3 were evaluated by averaging the results of 20 random repeats. From the figure, we can see that `TLRisk` always shows marked improvements compared with the baseline method `German Labels Only`, although there are still gaps between `TLRisk` and the ideal case `Lowerbound`. This indicates our algorithm `TLRisk` is effective on the cross-language classification problem.

| Data Set | 1 | 2 | 3 | 4 | 5 |
|---|---|---|---|---|---|
| German Labels Only | $0.246 \pm 0.061$ | $0.133 \pm 0.037$ | $0.301 \pm 0.067$ | $0.257 \pm 0.053$ | $0.277 \pm 0.068$ |
| TLRisk | $0.191 \pm 0.045$ | $0.122 \pm 0.043$ | $0.253 \pm 0.062$ | $0.247 \pm 0.059$ | $0.183 \pm 0.072$ |
| Lowerbound | $0.170 \pm 0.000$ | $0.116 \pm 0.000$ | $0.157 \pm 0.000$ | $0.176 \pm 0.000$ | $0.166 \pm 0.000$ |

Table 3: The average error rate and variance on each data set, given by all the evaluation methods, for English-to-German cross-language classification.

We have empirically tuned the trade-off parameter $\lambda$. Similar to the results of the text-aided image classification experiments, when $\lambda$ is small, the performance of `TLRisk` is better and stable. In

these experiments, we set $\lambda$ to $2^{-4}$. However, due to space limitation, we cannot present the curves for $\lambda$ tuning here.

## 5 Related Work

We review several prior works related to our work. To solve the label sparsity problem, researchers proposed several learning strategies, e.g. semi-supervised learning [13] and transfer learning [3, 11, 10, 9, 4]. Transfer learning mainly focuses on training and testing processes being in different scenarios, e.g. multi-task learning [3], learning with auxiliary data sources [11], learning from irrelevant categories [10], and self-taught learning [9, 4]. The *translated learning* proposed in this paper can be considered as an instance of general transfer learning; that is, transfer learning from data in different feature spaces.

Multi-view learning addresses learning across different feature spaces. Co-training [2] established the foundation of multi-view learning, in which the classifiers in two views learn from each other to enhance the learning process. Nigam and Ghani [8] proposed co-EM to apply EM algorithm to each view, and interchange probabilistic labels between different views. Co-EMT [7] is an active learning multi-view learning algorithm, and has shown more robustness empirically. However, as discussed before, multi-view learning requires that each instance should contain two views, while in translated learning, this requirement is relaxed. Translated learning can accept training data in one view and test data in another view.

## 6 Conclusions

In this paper, we proposed a translated learning framework for classifying target data using data from another feature space. We have shown that in translated learning, even though we have very little labeled data in the target space, if we can find a bridge to link the two spaces through feature translation, we can achieve good performance by leveraging the knowledge from the source data. We formally formulated our translated learning framework using risk minimization, and presented an approximation method for model estimation. In our experiments, we have demonstrated how this can be done effectively through the co-occurrence data in `TLRisk`. The experimental results on the text-aided image classification and the cross-language classification show that our algorithm can greatly outperform the state-of-the-art baseline methods.

**Acknowledgement** We thank the anonymous reviewers for their greatly helpful comments. Wenyuan Dai and Gui-Rong Xue are supported by the grants from National Natural Science Foundation of China (NO. 60873211) and the MSRA-SJTU joint lab project "Transfer Learning and its Application on the Web". Qiang Yang thanks the support of Hong Kong CERG Project 621307.

## Footnotes

[1] `http://www.vision.caltech.edu/Image_Datasets/Caltech256/`

[2]`http://www.ilovelanguages.com/idp/index.html`

## References

[1] N. Bel, C. Koster, and M. Villegas. Cross-lingual text categorization. In *ECDL*, 2003.

[2] A. Blum and T. Mitchell. Combining labeled and unlabeled data with co-training. In *COLT*, 1998.

[3] R. Caruana. Multitask learning. *Machine Learning*, 28(1):41–75, 1997.

[4] W. Dai, Q. Yang, G.-R. Xue, and Y. Yu. Self-taught clustering. In *ICML*, 2008.

[5] J. Lafferty and C. Zhai. Document language models, query models, and risk minimization for information retrieval. In *SIGIR*, 2001.

[6] D. Lowe. Distinctive image features from scale-invariant keypoints. *International Journal of Computer Vision*, 60(2):91–110, 2004.

[7] I. Muslea, S. Minton, and C. Knoblock. Active + semi-supervised learning = robust multi-view learning. In *ICML*, 2002.

[8] K. Nigam and R. Ghani. Analyzing the effectiveness and applicability of co-training. In *CIKM*, 2000.

[9] R. Raina, A. Battle, H. Lee, B. Packer, and A. Ng. Self-taught learning: transfer learning from unlabeled data. In *ICML*, 2007.

[10] R. Raina, A. Ng, and D. Koller. Constructing informative priors using transfer learning. In *ICML*, 2006.

[11] P. Wu and T. Dietterich. Improving svm accuracy by training on auxiliary data sources. In *ICML*, 2004.

[12] Y. Yang and J. Pedersen. A comparative study on feature selection in text categorization. In *ICML*, 1997.

[13] X. Zhu. Semi-supervised learning literature survey. Technical Report 1530, University of Wisconsin-Madison, 2007.
